# Optimal Movement Primitives

**Terence D. Sanger**
Jet Propulsion Laboratory
MS 303-310
4800 Oak Grove Drive
Pasadena, CA 91109
(818) 354-9127 tds@ai.mit.edu

## Abstract

The theory of Optimal Unsupervised Motor Learning shows how a network can discover a reduced-order controller for an unknown nonlinear system by representing only the most significant modes. Here, I extend the theory to apply to command sequences, so that the most significant components discovered by the network correspond to motion "primitives". Combinations of these primitives can be used to produce a wide variety of different movements. I demonstrate applications to human handwriting decomposition and synthesis, as well as to the analysis of electrophysiological experiments on movements resulting from stimulation of the frog spinal cord.

## 1 INTRODUCTION

There is much debate within the neuroscience community concerning the internal representation of movement, and current neurophysiological investigations are aimed at uncovering these representations. In this paper, I propose a different approach that attempts to define the optimal internal representation in terms of "movement primitives", and I compare this representation with the observed behavior. In this way, we can make strong predictions about internal signal processing. Deviations from the predictions can indicate biological constraints or alternative goals that cause the biological system to be suboptimal.

The concept of a motion primitive is not as well defined as that of a sensory primitive

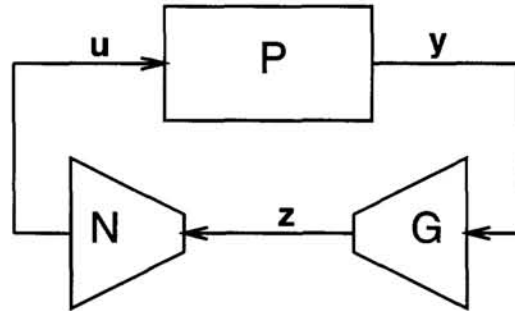

Figure 1: Unsupervised Motor Learning: The plant $P$ takes inputs $u$ and produces outputs $y$. The sensory map $G$ produces intermediate variables $z$, which are mapped onto the correct command inputs by the motor network $N$.

within the visual system, for example. There is no direct equivalent to the "receptive field" concept that has allowed interpretation of sensory recordings. In this paper, I will propose an internal model that involves both motor receptive fields and a set of movement primitives which are combined using a weighted sum to produce a large class of movements. In this way, a small number of well-designed primitives can generate the full range of desired behaviors.

I have previously developed the concept of "optimal unsupervised motor learning" to investigate optimal internal representations for instantaneous motor commands. The optimal representations adaptively discover a reduced-order linearizing controller for an unknown nonlinear plant. The theorems give the optimal solution in general, and can be applied to special cases for which both linear and nonlinear adaptive algorithms exist (Sanger 1994b). In order to apply the theory to complete movements it needs to be extended slightly, since in general movements exist within an infinite-dimensional task space rather than a finite-dimensional control space. The goal is to derive a small number of primitives that optimally encode the full set of observed movements. Generation of the internal movement primitives then becomes a data-compression problem, and I will choose primitives that minimize the resultant mean-squared error.

## 2    OPTIMAL UNSUPERVISED MOTOR LEARNING

Optimal Unsupervised Motor Learning is based on three principles:

1. Dimensionality Reduction
2. Accurate Reduced-order Control
3. Minimum Sensory error

Consider the system shown in figure 1. At time $t$, the plant $P$ takes motor inputs $u$ and produces sensory outputs $y$. A sensory mapping $G$ transforms the raw sensory data $y$ to an intermediate representation $z$. A motor mapping takes desired values of $z$ and computes the appropriate command $u$ such that $GPu = z$. Note that the

loop in the figure is not a feedback-control loop, but is intended to indicate the flow of information. With this diagram in mind, we can write the three principles as:

1. $\dim[z] < \dim[y]$
2. $GPNz = z$
3. $\|PNGy - y\|$ is minimized

We can prove the following theorems (Sanger 1994b):

*Theorem 1:* For all $G$ there exists an $N$ such that $GPNz = z$. If $G$ is linear and $P^{-1}$ is linear, then $N$ is linear.

*Theorem 2:* For any $G$, define an invertible map $\bar{G}$ such that $G\bar{G}^{-1} = I$ on range$[G]$. Then $\|PNGy - y\|$ is minimized when $G$ is chosen such that $\|y - \bar{G}^{-1}G\|$ is minimized. If $G$ and $P$ are linear and the singular value decomposition of $P$ is given by $L^TSR$, then the optimal maps are $G = L$ and $N = R^TS^{-1}$.

For the discussion of movement, the linear case will be the most important since in the nonlinear case we can use unsupervised motor learning to perform dimensionality reduction and linearization of the plant at each time $t$. The movement problem then becomes an infinite-dimensional linear problem.

Previously, I have developed two iterative algorithms for computing the singular value decomposition from input/output samples (Sanger 1994a). The algorithms are called the "Double Generalized Hebbian Algorithm" (DGHA) and the "Orthogonal Asymmetric Encoder" (OAE). DGHA is given by

$$\begin{aligned} \Delta G &= \gamma(zy^T - \mathrm{LT}[zz^T]G) \\ \Delta N^T &= \gamma(zu^T - \mathrm{LT}[zz^T]N^T) \end{aligned}$$

while OAE is described by:

$$\begin{aligned} \Delta G &= \gamma(\hat{z}y^T - \mathrm{LT}[\hat{z}\hat{z}^T]G) \\ \Delta N^T &= \gamma(Gy - \mathrm{LT}[GG^T]\hat{z})u^T \end{aligned}$$

where LT[ ] is an operator that sets the above diagonal elements of its matrix argument to zero, $y = Pu$, $z = Gy$, $\hat{z} = N^Tu$, and $\gamma$ is a learning rate constant. Both algorithms cause $G$ to converge to the matrix of left singular vectors of $P$, and $N$ to converge to the matrix of right singular vectors of $P$ (multiplied by a diagonal matrix for DGHA). DGHA is used in the examples below.

## 3 MOVEMENT

In order to extend the above discussion to allow adaptive discovery of movement primitives, we now consider the plant $P$ to be a mapping from command sequences $u(t)$ to sensory sequences $y(t)$. We will assume that the plant has been feedback linearized (perhaps by unsupervised motor learning). We also assume that the sensory network $G$ is constrained to be linear. In this case, the optimal motor network $N$ will also be linear. The intermediate variables $z$ will be represented by a vector. The sensory mapping consists of a set of sensory "receptive fields" $g_i(t)$

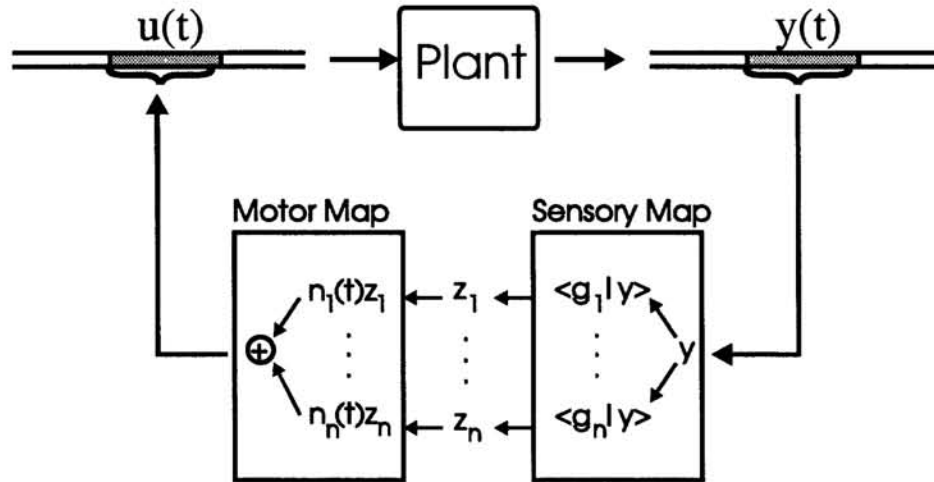

Figure 2: Extension of unsupervised motor learning to the case of trajectories. Plant input and output are time-sequences $u(t)$ and $y(t)$. The sensory and motor maps now consist of sensory primitives $g_i(t)$ and motor primitives $n_i(t)$.

such that

$$z_i = \int g_i(t)y(t)dt = < g_i|y >$$

and the motor mapping consists of a set of "motor primitives" $n_i(t)$ such that

$$u(t) = \sum_i n_i(t)z_i$$

as in figure 2. If the plant is equal to the identity (complete feedback linearization), then $g_i(t) = n_i(t)$. In this case, the optimal sensory-motor primitives are given by the eigenfunctions of the autocorrelation function of $y(t)$. If the autocorrelation is stationary, then the infinite-window eigenfunctions will be sinusoids. Note that the optimal primitives depend both on the plant $P$ as well as the statistical distribution of outputs $y(t)$.

In practice, both $u(t)$ and $y(t)$ are sampled at discrete time-points $\{t_k\}$ over a finite time-window, so that the plant input and output is in actuality a long vector. Since the plant is linear, the optimal solution is given by the singular value decomposition, and either the DGHA or OAE algorithms can be used directly. The resulting sensory primitives map the sensory information $y(t)$ onto the finite-dimensional $z$, which is usually a significant data compression. The motor primitives map $z$ onto the sequence $u(t)$, and the resulting $\hat{y}(t) = P[u(t)]$ will be a linear projection of $y(t)$ onto the space spanned by the set $\{Pn_i(t)\}$.

## 4   EXAMPLE 1: HANDWRITING

As a simple illustration, I examine the case of human handwriting. We can consider the plant to be the identity mapping from pen position to pen position, and the

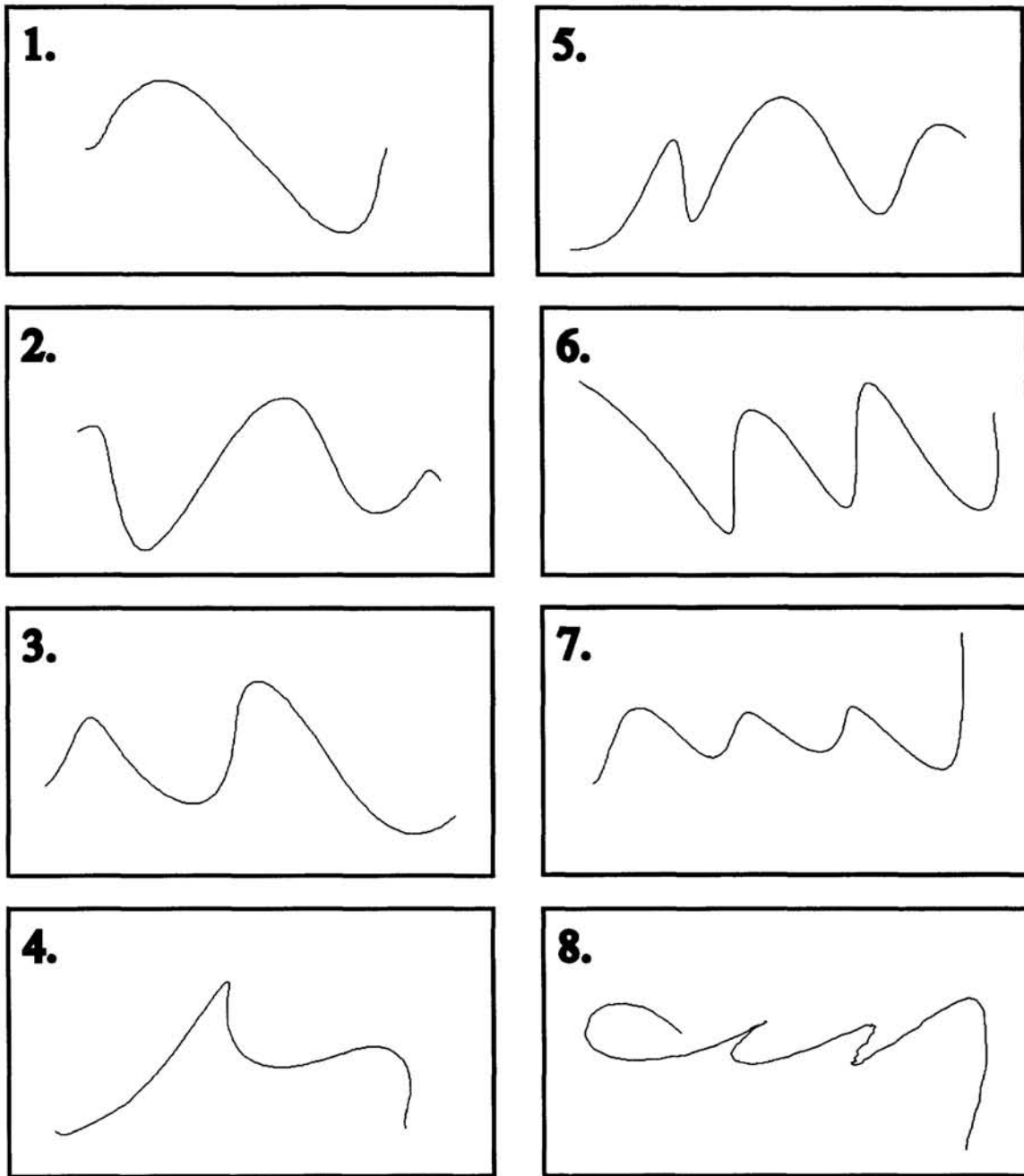

Figure 3: Movement primitives for sampled human handwriting.

human to be taking desired sensory values of pen position and converting them into motor commands to move the pen. The sensory statistics then reflect the set of trajectories used in producing handwritten letters. An optimal reduced-order control system can be designed based on the observed statistics, and its performance can be compared to human performance.

For this example, I chose sampled data from 87 different examples of lower-case letters written by a single person, and represented as horizontal and vertical pen position at each point in time. Blocks of 128 sequential points were used for training, and 8 internal variables $z_i$ were used for each of the two components of pen position. The training set consisted of 5000 randomly chosen samples. Since the plant is the identity, the sensory and motor primitives are the same, and these are shown as "strokes" in figure 3. Linear combinations of these strokes can be used to generate pen paths for drawing lowercase letters. This is shown in figure 4, where the word "hello" (not present in the training set) is written and projected using increasing numbers of intermediate variables $z_i$. The bottom of figure 4 shows the sequence of values of $z_i$ that was used (horizontal component only).

Good reproduction of the test word was achieved with 5 movement primitives. A total of 7 128-point segments was projected, and these were recombined using smooth 50% overlap. Each segment was encoded by 5 coefficients for each of the horizontal and vertical components, giving a total of 70 coefficients to represent 1792 data points (896 horizontal and vertical components), for a compression ratio of 25:1.

# 5   EXAMPLE 2: FROG SPINAL CORD

The second example models some interesting and unexplained neurophysiological results from microstimulation of the frog spinal cord. (Bizzi *et al.* 1991) measured the pattern of forces produced by the frog hindlimb at various positions in the workspace during stimulation of spinal interneurons. The resulting force-fields often have a stable "equilibrium point", and in some cases this equilibrium point follows a smooth closed trajectory during tonic stimulation of the interneuron. However, only a small number of different force field shapes have been found, and an even smaller number of different trajectory types. A hypothesis to explain this result is that larger classes of different trajectories can be formed by combining the patterns produced by these cells. This hypothesis can be modelled using the optimal movement primitives described above.

Figure 5a shows a simulation of the frog leg. To train the network, random smooth planar movements were made for 5000 time points. The plant output was considered to be 32 successive cartesian endpoint positions, and the plant input was the time-varying force vector field. Two hidden units $z$ were used. In figure 5b we see an example of the two equilibrium point trajectories (movement primitives) that were learned by DGHA. Linear combinations of these trajectories account for over 96% of the variance of the training data, and they can approximate a large class of smooth movements. Note that many other pairs of orthogonal trajectories can accomplish this, and different trials often produced different orthogonal trajectory shapes.

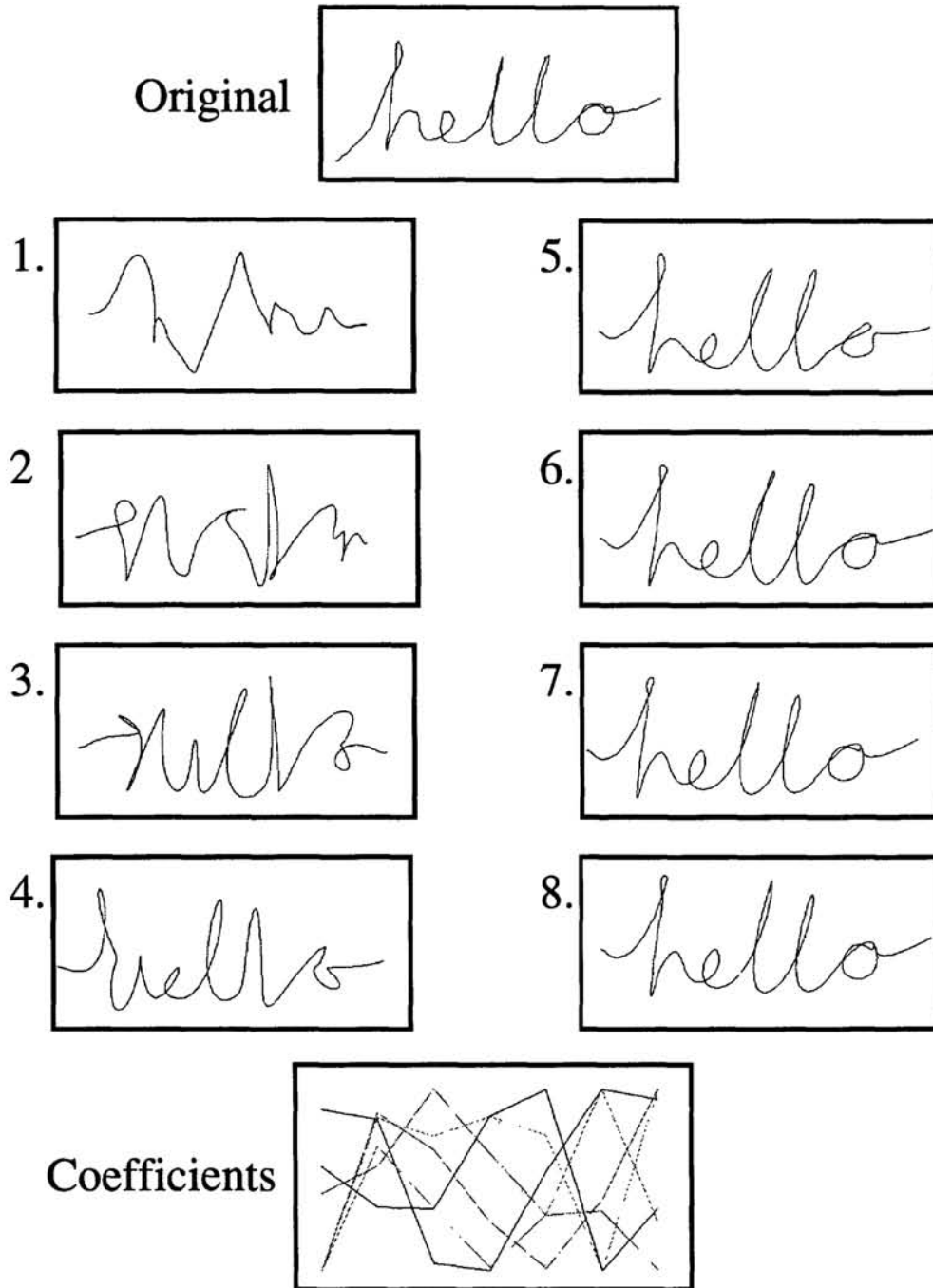

Figure 4: Projection of test-word "hello" using increasing numbers of intermediate variables $z_i$.

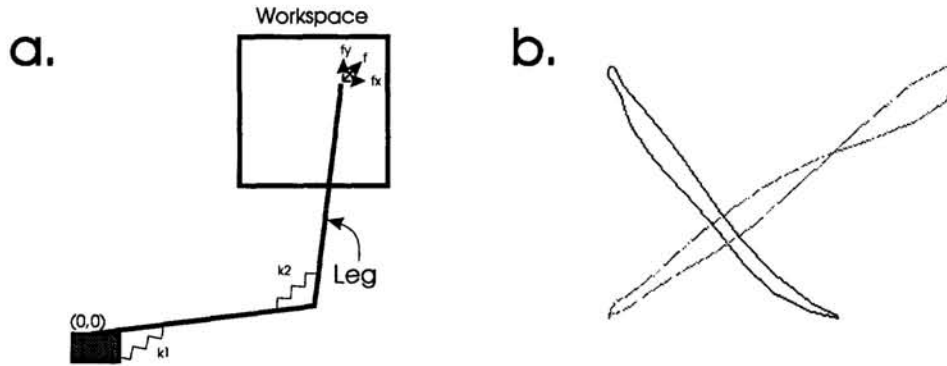

Figure 5: a. Simulation of frog leg configuration. b. An example of learned optimal movement primitives.

# 6   CONCLUSION

The examples are not meant to provide detailed models of internal processing for human or frog motor control. Rather, they are intended to illustrate the concept of optimal primitives and perhaps guide the search for neurophysiological and psychophysical correlates of these primitives. The first example shows that generation of the lower-case alphabet can be accomplished with approximately 10 coefficients per letter, and that this covers a considerable range of variability in character production. The second example demonstrates that an adaptive algorithm allows the possibility for the frog spinal cord to control movement using a very small number of internal variables.

Optimal unsupervised motor learning thus provides a descriptive model for the generation of a large class of movements using a compressed internal description. A set of fixed movement primitives can be combined linearly to produce the necessary motor commands, and the optimal choice of these primitives assures that the error in the resulting movement will be minimized.

## References

Bizzi E., Mussa-Ivaldi F. A., Giszter S., 1991, Computations underlying the execution of movement: A biological perspective, *Science*, 253:287–291.

Sanger T. D., 1994a,  Two algorithms for iterative computation of the singular value decomposition from input/output samples, In Touretzky D., ed., *Advances in Neural Information Processing 6*, Morgan Kaufmann, San Mateo, CA, in press.

Sanger T. D., 1994b, Optimal unsupervised motor learning, *IEEE Trans. Neural Networks*, in press.
